# Spectral Kernel Methods for Clustering

**Nello Cristianini**
BIOwulf Technologies
*nello@support-vector.net*

**John Shawe-Taylor**   **Jaz Kandola**
Royal Holloway, University of London
*{john, jaz}@cs.rhul.ac.uk*

## Abstract

In this paper we introduce new algorithms for unsupervised learning based on the use of a kernel matrix. All the information required by such algorithms is contained in the eigenvectors of the matrix or of closely related matrices. We use two different but related cost functions, the Alignment and the 'cut cost'. The first one is discussed in a companion paper [3], the second one is based on graph theoretic concepts. Both functions measure the level of clustering of a labeled dataset, or the correlation between data clusters and labels. We state the problem of unsupervised learning as assigning labels so as to optimize these cost functions. We show how the optimal solution can be approximated by slightly relaxing the corresponding optimization problem, and how this corresponds to using eigenvector information. The resulting simple algorithms are tested on real world data with positive results.

## 1   Introduction

Kernel based learning provides a modular approach to learning system design [2]. A general algorithm can be selected for the appropriate task before being mapped onto a particular application through the choice of a problem specific kernel function.

The kernel based method works by mapping data to a high dimensional feature space implicitly defined by the choice of the kernel function. The kernel function computes the inner product of the images of two inputs in the feature space. From a practitioners viewpoint this function can also be regarded as a similarity measure and hence provides a natural way of incorporating domain knowledge about the problem into the bias of the system.

One important learning problem is that of dividing the data into classes according to a cost function together with their relative positions in the feature space. We can think of this as clustering in the kernel defined feature space, or non-linear clustering in the input space.

In this paper we introduce two novel kernel-based methods for clustering. They both assume that a kernel has been chosen and the kernel matrix constructed. The methods then make use of the matrix's eigenvectors, or of the eigenvectors of the closely related Laplacian matrix, in order to infer a label assignment that approximately optimizes one of two cost functions. See also [4] for use of spectral decompositions of the kernel matrix. The paper includes some analysis of the algorithms together with tests of the methods on real world data with encouraging results.

## 2  Two partition cost measures

All the information needed to specify a clustering of a set of data is contained in the matrix $M_{ij} = (\text{cluster}(x_i) == \text{cluster}(x_j))$, where $(A == B) \in \{-1, +1\}$. After a clustering is specified, one can measure its cost in many ways. We propose here two cost functions that are easy to compute and lead to efficient algorithms.

Learning is possible when some collusion between input distribution and target exists, so that we can predict the target based on the input. Typically one would expect points with similar labels to be clustered and the clusters to be separated.

This can be detected in two ways: either by measuring the amount of label-clustering or by measuring the correlation between such variables. In the first case, we need to measure how points of the same class are close to each other and distant from points of different classes. In the second case, kernels can be regarded as oracles predicting whether two points are in the same class. The 'true' oracle is the one that knows the true matrix $M$. A measure of quality can be obtained by measuring the Pearson correlation coefficient between the kernel matrix $K$ and the true $M$.

Both approaches lead to the same quantity, known as the alignment [3].

We will use the following definition of the inner product between matrices $\langle K_1, K_2 \rangle_F = \sum_{i,j=1}^{m} K_1(x_i, x_j) K_2(x_i, x_j)$. The index $F$ refers to the Frobenius norm that corresponds to this inner product.

**Definition 1 Alignment** *The (empirical) alignment of a kernel $k_1$ with a kernel $k_2$ with respect to the sample $S$ is the quantity*

$$\hat{A}(S, k_1, k_2) = \frac{\langle K_1, K_2 \rangle_F}{\sqrt{\langle K_1, K_1 \rangle_F \langle K_2, K_2 \rangle_F}},$$

*where $K_i$ is the kernel matrix for the sample $S$ using kernel $k_i$.*

This can also be viewed as the cosine of the angle between to bi-dimensional vectors $K_1$ and $K_2$, representing the Gram matrices. If we consider $k_2 = yy'$, where $y$ is the vector of $\{-1, +1\}$ labels for the sample, then with a slight abuse of notation

$$\hat{A}(S, k, y) = \frac{\langle K, yy' \rangle_F}{\sqrt{\langle K, K \rangle_F \langle yy', yy' \rangle_F}} = \frac{\langle K, yy' \rangle_F}{m \|K\|_F}, \text{ since } \langle yy', yy' \rangle_F = m^2$$

Another measure of separation between classes is the average separation between two points in different classes, again normalised by the matrix norm.

**Definition 2 Cut Cost.** *The cut cost of a clustering is defined as*

$$C(S, k, y) = \frac{\sum_{ij: y_i \neq y_j} k(x_i, x_j)}{m \|K\|_F}.$$

This quantity is motivated by a graph theoretic concept. If we consider the Kernel matrix as the adjacency matrix of a fully connected weighted graph whose nodes are the data points, the cost of partitioning a graph is given by the total weight of the edges that one needs to cut or remove, and is exactly the numerator of the 'cut cost'. Notice also the relation between alignment and cutcost:

$$\hat{A}(S, k, y) = \frac{\sum_{ij} k(x_i, x_j) - 2C(S, k)}{m \sqrt{\langle K, K \rangle_F}} = T(S, k) - 2C(S, k, y),$$

where $T(S, k) = \hat{A}(S, k, j)$, for $j$ the all ones vector. Among other appealing properties of the alignment, is that this quantity is sharply concentrated around

its mean, as proven in the companion paper [3]. This shows that the expected alignment can be reliably estimated from its empirical estimate $\hat{A}(S)$. As the cut cost can be expressed as the difference of two alignments

$$C(S, k, y) = 0.5(T(S, k) - \hat{A}(S, k, y)),\qquad(1)$$

it will be similarly concentrated around its expected value.

## 3 Optimising the cost with spectral techniques

In this section we will introduce and test two related methods for clustering, as well as their extensions to transduction. The general problem we want to solve is to assign class-labels to datapoints so as to maximize one of the two cost functions given above. By equation (1) the optimal solution to both problems is identical for a fixed data set and kernel. The difference between the approaches is in the two approximation algorithms developed for the different cost functions. The approximation algorithms are obtained by relaxing the discrete problems of optimising over all possible labellings of a dataset to closely related continuous problems solved by eigenvalue decompositions. See [5] for use of eigenvectors in partitioning sparse matrices.

### 3.1 Optimising the alignment

To optimise the alignment, the problem is to find the maximally aligned set of labels

$$\hat{A}^*(S, k) = \max_{y \in \{-1, 1\}^m} \hat{A}(S, k, y) = \max_{y \in \{-1, 1\}^m} \frac{\langle K, yy' \rangle_F}{m\sqrt{\langle K, K \rangle_F}}$$

Since in this setting the kernel is fixed maximising the alignment reduces to choosing $y \in \{-1, 1\}^m$ to maximise $\langle K, yy' \rangle = y'Ky$. If we allow $y$ to be chosen from the larger set $\mathbb{R}^m$ subject to the constraint $\|y\|^2 = m$, we obtain an approximate maximum-alignment problem that can be solved efficiently. After solving the relaxed problem, we can obtain an approximate discrete solution by choosing a suitable threshold to the entries in the vector $y$ and applying the sign function. Bounds will be given on the quality of the approximations.

The solution of the approximate problem follows from the following theorem that provides a variational characterization of the spectrum of symmetric matrices.

**Theorem 3 (Courant-Fischer Minimax Theorem)** *If $M \in \mathbb{R}^{m \times m}$ is symmetric, then for $k = 1, \ldots, m$,*

$$\lambda_k(M) = \max_{\dim(T)=k} \min_{0 \neq v \in T} \frac{v'Mv}{v'v} = \min_{\dim(T)=m-k+1} \max_{0 \neq v \in T} \frac{v'Mv}{v'v},$$

If we consider the first eigenvector, the first min does not apply and we obtain that the approximate alignment problem is solved by the first eigenvector, so that the maximal alignment is upper bounded by a multiple of the first eigenvalue, $\lambda_{\max} = \max_{0 \neq v \in \mathbb{R}^m} \frac{v'Kv}{v'v}$. One can now transform the vector $v$ into a vector in $\{-1, +1\}^m$ by choosing the threshold $\theta$ that gives maximum alignment of $y = \text{sign}(v^{\max} - \theta)$.

By definition, the value of alignment $\hat{A}(S, k, y)$ obtained by this $y$ will be a lower bound of the optimal alignment, hence we have

$$\hat{A}(S, k, y) \leq \hat{A}^*(S, k) \leq \lambda_{\max}/\|K\|_F.$$

One can hence estimate the quality of a dichotomy by comparing its value with the upper bound. The absolute alignment tells us how specialized a kernel is on a given dataset: the higher this quantity, the more committed to a specific dichotomy.

The first eigenvector can be calculated in many ways, for example the Lanczos procedure, which is already effective for large datasets. Search engines like Google are based on estimating the first eigenvector of a matrix with dimensionality more than $10^9$, so for very large datasets there are approximation techniques.

We applied the procedure outlined above to two datasets from the UCI repository. We preprocessed the data by normalising the input vectors in the kernel defined feature space and then centering them by shifting the origin (of the feature space) to their centre of gravity. This can be achieved by the following transformation of the kernel matrix, $K \longleftarrow K - m^{-1}jg' - m^{-1}gj' + m^{-2}j'KjJ$, where $j$ is the all ones vector, $J$ the all ones matrix and $g$ the vector of row sums of $K$.

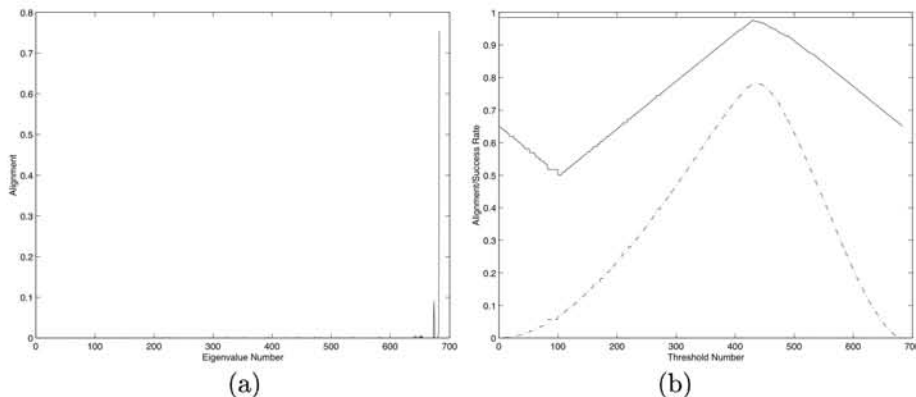

(a)                                        (b)

Figure 1: (a) Plot of alignment of the different eigenvectors with the labels ordered by increasing eigenvalue. (b) Plot for Breast Cancer data (linear kernel) of $\lambda_{max}/\|K\|_F$ (straight line), $\hat{A}(S,k,y)$ for $y = \text{sign}(v^{max} - \theta_i)$ (bottom curve), and the accuracy of $y$ (middle curve) against threshold number $i$.

The first experiment applied the unsupervised technique to the Breast Cancer data with a linear kernel. Figure 1(a) shows the alignmment of the different eigenvectors with the labels. The highest alignment is shown by the last eigenvector corresponding to the largest eigenvalue.

For each value $\theta_i$ of the threshold Figure 1(b) shows the upper bound of $\lambda_{max}/\|K\|_F$ (straight line), the alignment $\hat{A}(S,k,y)$ for $y = \text{sign}(v^{max} - \theta_i)$ (bottom curve), and the accuracy of $y$ (middle curve). Notice that where actual alignment and upper bound on alignment get closest, we have confidence that we have partitioned our data well, and in fact the accuracy is also maximized. Notice also that the choice of the threshold corresponds to maintaining the correct proportion between positives and negatives. This suggests another possible threshold selection strategy, based on the availability of enough labeled points to give a good estimate of the proportion of positive points in the dataset. This is one way label information can be used to choose the threshold. At the end of the experiments we will describe another 'transduction' method.

It is a measure of how naturally the data separates that this procedure is able to optimise the split with an accuracy of approximately 97.29% by choosing the threshold that maximises the alignment (threshold number 435) but *without making any use of the labels*.

In Figure 2a we present the same results for the Gaussian kernel ($\sigma = 6$). In this case the accuracy obtained by optimising the alignment (threshold number 316) of the resulting dichotomy is less impressive being only about 79.65%. Finally, Figure 2b shows the same results for the Ionosphere dataset. Here the accuracy of the split that optimises the alignment (threshold number 158) is approximately

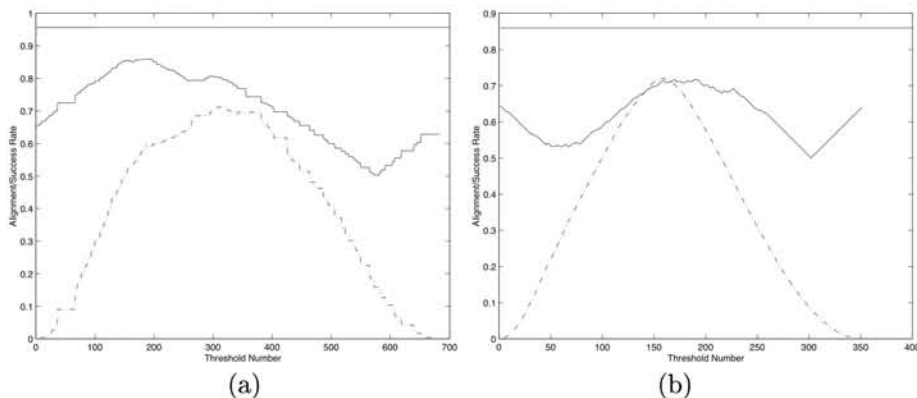

<center>(a)                                         (b)</center>

Figure 2: Plot for Breast Cancer data (Gaussian kernel) (a) and Ionosphere data (linear kernel) (b) of $\lambda_{max}/\|K\|_F$ (straight line), $\hat{A}(S,k,y)$ for $y = \mathrm{sign}(v^{max} - \theta_i)$ (bottom curve), and the accuracy of $y$ (middle curve) against threshold number $i$.

71.37%.

We can also use the overall approach to adapt the kernel to the data. For example we can choose the kernel parameters so as to optimize $\lambda_{\max}/\|K\|_F$. Then find the first eigenvector, choose a threshold to maximise the alignment and output the corresponding $y$.

The cost to the alignment of changing a label $y_i$ is $2\sum_j y_j k(x_i, x_j)/\|K\|_F$, so that if a point is isolated from the others, or if it is equally close to the two different classes, then changing its label will have only a very small effect. On the other hand, labels in strongly clustered points clearly contribute to the overall cost and changing their label will alter the alignment significantly.

The method we have described can be viewed as projecting the data into a 1-dimensional space and finding a threshold. The projection also implicitly sorts the data so that points of the same class are nearby in the ordering. We discuss the problem in the 2-class case. We consider embedding the set into the real line, so as to satisfy a clustering criterion. The resulting Kernel matrix should appear as a block diagonal matrix.

This problem has been addressed in the case of information retrieval in [1], and also applied to assembling sequences of DNA. In those cases, the eigenvectors of the Laplacian have been used, and the approach is called the Fiedler ordering. Although the Fiedler ordering could be used here as well, we present here a variation based on the simple kernel matrix.

Let the coordinate of the point $x_i$ on the real line be $v(i)$. Consider the cost function $\sum_{ij} v(i)v(j)K(i,j)$. It is maximized when points with high similarity have the same sign and high absolute value, and when points with different sign have low similarity.

The choice of coordinates $v$ that optimizes this cost is the first eigenvector, and hence by sorting the data according to the value of their entry in this eigenvector one can hope to find a good permutation, that renders the kernel matrix block diagonal. Figure 3 shows the results of this heuristic applied to the Breast cancer dataset. The grey level indicates the size of the kernel entry. The figure on the left is for the unsorted data, while that on the right shows the same plot after sorting. The sorted figure clearly shows the effectivenesss of the method.

## 3.2   Optimising the cut-cost

For a fixed kernel matrix minimising the *cut-cost* corresponds to minimising $\sum_{y_i \neq y_j} k(x_i, x_j)$, that is the sum of the kernel entries between points of two dif-

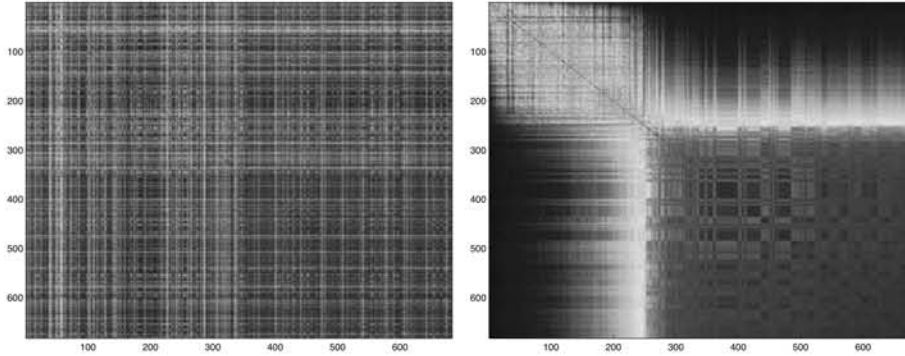

Figure 3: Gram matrix for cancer data, before and after permutation of data according to sorting order of first eigenvector of $K$

ferent classes. Since we are dealing with normalized kernels, this also controls the expected distance between them.

We can express this quantity as
$$\sum_{y_i \neq y_j} K_{ij} = \frac{1}{2} \left( \sum_{i,j} K_{ij} - y'Ky \right) = \frac{1}{2} y'Ly,$$

where $L$ is the Laplacian matrix, defined as $L = D - K$, where $D = \text{diag}(d_1, \ldots, d_m)$ with $d_i = \sum_{j=1}^{m} k(x_i, x_j)$. One would like to find $y \in \{-1, +1\}^m$ so as to minimize the cut cost subject to the division being even, but this problem is NP-hard. Following the same strategy as with the alignment we can impose a slightly looser constraint on $y$, $y \in \mathbb{R}^m$, $\sum_i y_i^2 = m$, $\sum_i y_i = 0$. This gives the problem

$$\min y'Ly \quad \text{subject to } y \in \mathbb{R}^m, \sum_i y_i^2 = m, \sum_i y_i = 0.$$

Since, zero is an eigenvalue of $L$ with eigenvector $j$, the all ones vector, the problem is equivalent to finding the eigenvector of the smallest non-zero eigenvalue $\lambda = \min_{0 \neq y \perp j} \frac{y'Ly}{y'y}$. Hence, this eigenvalue $\lambda$ provides a lower bound on the cut cost

$$\min_{y \in \{-1, 1\}^m} C(S, k, y) \geq \frac{\lambda}{2\|K\|_F}.$$

So the eigenvector corresponding to the eigenvalue $\lambda$ of the Laplacian can be used to obtain a good approximate split and $\lambda$ gives a lower bound on the cut-cost. One can now threshold the entries of the eigenvector in order to obtain a vector with $-1$ and $+1$ entries. We again plot the lower bound, cut-cost, and error rate as a function of the threshold.

We applied the procedure to the Breast cancer data with both linear and Gaussian kernels. The results are shown in Figure 4. Now using the cut cost to select the best threshold for the linear kernel sets it at 378 with an accuracy of 67.86%, significantly worse than the results obtained by optimising the alignment. With the Gaussian kernel, on the other hand, the method selects threshold 312 with an accuracy of 80.31%, a slight improvement over the results obtained with this kernel by optimising the alignment.

So far we have presented algorithms that use unsupervised data. We now consider the situation where we are given a partially labelled dataset. This leads to a simple algorithm for transduction or semi-supervised learning. The idea that some labelled data might improve performance comes from observing Figure 4b, where the selection based on the cut-cost is clearly suboptimal. By incorporating some label information, it is hoped that we can obtain an improved threshold selection.

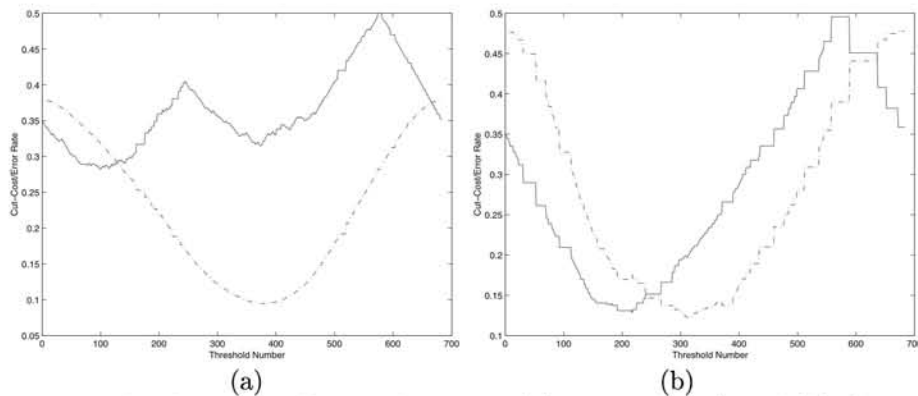

<div align="center">(a)                 (b)</div>

Figure 4: Plot for Breast Cancer data using (a) Linear kernel) and (b) Gaussian kernel of $C(S, k, y) - \lambda/(2\|K\|_F)$ (dashed curves), for $y = \mathrm{sign}(v^{max} - \theta_i)$ and the error of $y$ (solid curve) against threshold number $i$.

Let $z$ be the vector containing the known labels and 0 elsewhere. Set $KP = K + C_0 z z'$, where $C_0$ is a positive constant parameter. We now use the original matrix $K$ to generate the eigenvector, but the matrix $KP$ when measuring the cut-cost of the classifications generated by different thresholds. Taking $C_0 = 1$ we performed 5 random selections of 20% of the data and obtained a mean success rate of 85.56% (standard deviation 0.67%) for the Breast cancer data with Gaussian kernel, a marked improvement over the 80.31% achieved with no label information.

## 4 Conclusions

The paper has considered two partition costs the first derived from the so-called alignment of a kernel to a label vector, and the second from the cut-cost of a label vector for a given kernel matrix. The two quantities are both optimised by the same labelling, but give rise to different approximation algorithms when the discrete constraint is removed from the labelling vector. It was shown how these relaxed problems are solved exactly using spectral techniques, hence leading to two distinct approximation algorithms through a post-processing phase that re-discretises the vector to create a labelling that is chosen to optimise the given criterion.

Experiments are presented showing the performance of both of these clustering techniques with some very striking results. For the second algorithm we also gave one preliminary experiment with a transductive version that enables some labelled data to further refine the clustering.

## References

[1] M.W. Berry, B. Hendrickson, and P. Raghavan. Sparse matrix reordering schemes for browsing hypertext. In *The Matematics of Numerical Analysis*, pages 99–123. AMS, 1996.

[2] N. Cristianini and J. Shawe-Taylor. *An Introduction to Support Vector Machines.* Cambridge University Press, 2000. See also the web site **www.support-vector.net**.

[3] Nello Cristianini, André Elisseeff, John Shawe-Taylor, and Jaz Kandola. On kernel-target alignment. In *submitted to Proceedings of Neural Information Processing Systems (NIPS)*, 2001.

[4] Nello Cristianini, Huma Lodhi, and John Shawe-Taylor. Latent semantic kernels for feature selection. Technical Report NC-TR-00-080, NeuroCOLT Working Group, http://www.neurocolt.org, 2000.

[5] A. Pothen, H. Simon, and K. Liou. Partitioning sparse matrices with eigenvectors of graphs. *SIAM J. Matrix Anal.*, 11(3):430–452, 1990.
